# NEURAL CONTROL OF SENSORY ACQUISITION: THE VESTIBULO-OCULAR REFLEX.

Michael G. Paulin, Mark E. Nelson and James M. Bower
Division of Biology
California Institute of Technology
Pasadena, CA 91125

## ABSTRACT

We present a new hypothesis that the cerebellum plays a key role in actively controlling the acquisition of sensory information by the nervous system. In this paper we explore this idea by examining the function of a simple cerebellar-related behavior, the vestibulo-ocular reflex or VOR, in which eye movements are generated to minimize image slip on the retina during rapid head movements. Considering this system from the point of view of statistical estimation theory, our results suggest that the transfer function of the VOR, often regarded as a static or slowly modifiable feature of the system, should actually be continuously and rapidly changed during head movements. We further suggest that these changes are under the direct control of the cerebellar cortex and propose experiments to test this hypothesis.

## 1. INTRODUCTION

A major thrust of research in our laboratory involves exploring the way in which the nervous system actively controls the acquisition of information about the outside world. This emphasis is founded on our suspicion that the principal role of the cerebellum, through its influence on motor systems, is to monitor and optimize the quality of sensory information entering the brain. To explore this question, we have undertaken an investigation of the simplest example of a cerebellar-related motor activity that results in improved sensory inputs, the vestibulo-ocular reflex (VOR). This reflex is responsible for moving the eyes to compensate for rapid head movements to prevent retinal image slip which would otherwise significantly degrade visual acuity (Carpenter, 1977).

## 2. VESTIBULO-OCULAR REFLEX (VOR)

The VOR relies on the vestibular apparatus of the inner ear which is an inertial sensor that detects movements of the head. Vestibular output caused by head movements give rise to compensatory eye movements through an anatomically well described neural pathway in the brain stem (for a review see Ito, 1984). Visual feedback also makes an important contribution to compensatory eye movements during slow head movements,

but during rapid head movements with frequency components greater than about 1Hz, the vestibular component dominates (Carpenter, 1977).

A simple analysis of the image stabilization problem indicates that during head rotation in a single plane, the eyes should be made to rotate at equal velocity in the opposite direction. This implies that, in a simple feedforward control model, the VOR transfer function should have unity gain and a 180° phase shift. This would assure stabilized retinal images of distant objects. It turns out, however, that actual measurements reveal the situation is not this simple. Furman, O'Leary and Wolfe (1982), for example, found that the monkey VOR has approximately unity gain and 180° phase shift only in a narrow frequency band around 2Hz. At 4Hz the gain is too high by a factor of about 30% (fig. 1).

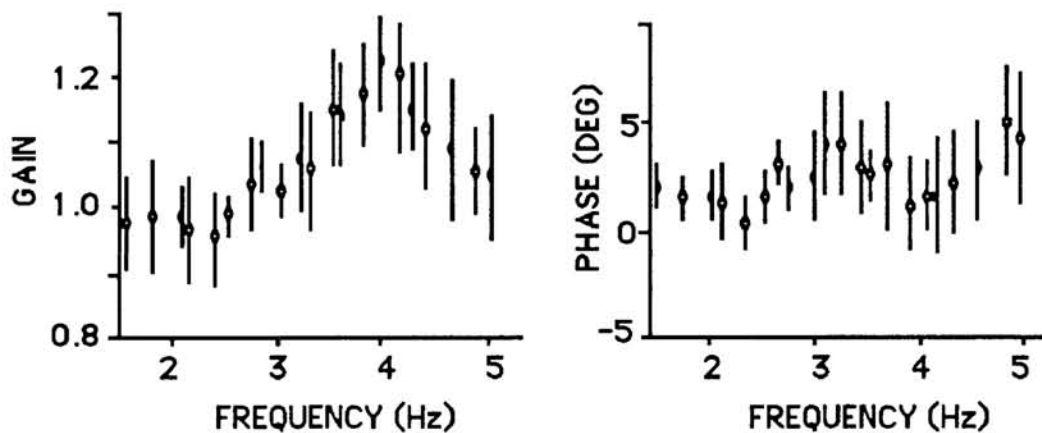

Figure 1: Bode gain and phase plots for the transfer function of the horizontal component of the VOR of the alert Rhesus monkey at high frequencies (Data from Furman et al. (1982)).

Given the expectation of unity gain, one might be tempted to conclude from the monkey data that the VOR simply does not perform well at high frequencies. But 4Hz is not a very high frequency for head movements, and perhaps it is not the VOR which is performing poorly, but the simplified analysis using classical control theory. In this paper, we argue that the VOR uses a more sophisticated strategy and that the "excessive" gain in the system seen at higher frequencies actually improves VOR performance.

## 3. OPTIMAL ESTIMATION

In order to understand the discrepancy between the predictions of simple control theory models and measured VOR dynamics, we believe it is necessary to take into account more of the real world conditions under which the VOR operates. Examples include noisy head velocity measurements, conduction delays and multiple, possibly conflicting, measurements of head velocity, acceleration, muscle contractions, etc., generated by different sensory modalities. The mathematical framework that is appropriate for analyz-

ing problems of this kind is stochastic state-space dynamical systems theory (Davis and Vinter, 1985). This framework is an extension of classical linear dynamical systems theory that accommodates multiple inputs and outputs, nonlinearities, time-varying dynamics, noise and delays. One area of application of the state space theory has been in target tracking, where the basic principle involves using knowledge of the dynamics of a target to estimate its most probable trajectory given imprecise data. The VOR can be viewed as a target tracking system whose target is the "world", which moves in head coordinates. We have reexamined the VOR from this point of view.

**The Basic VOR.**
To begin our analysis of the VOR we have modeled the eye-head-neck system as a damped inverted pendulum with linear restoring forces (fig. 2) where the model system is driven by random (Gaussian white) torque. Within this model, we want to predict the correct compensatory "eye" movements during "head" movements to stabilize the direction in which the eye is pointing. Figure 2 shows the amplitude spectrum of head velocity for this model. In this case, the parameters of the model result in a system that has a natural resonance in the range of 1 to 2 Hz and attenuates higher frequencies.

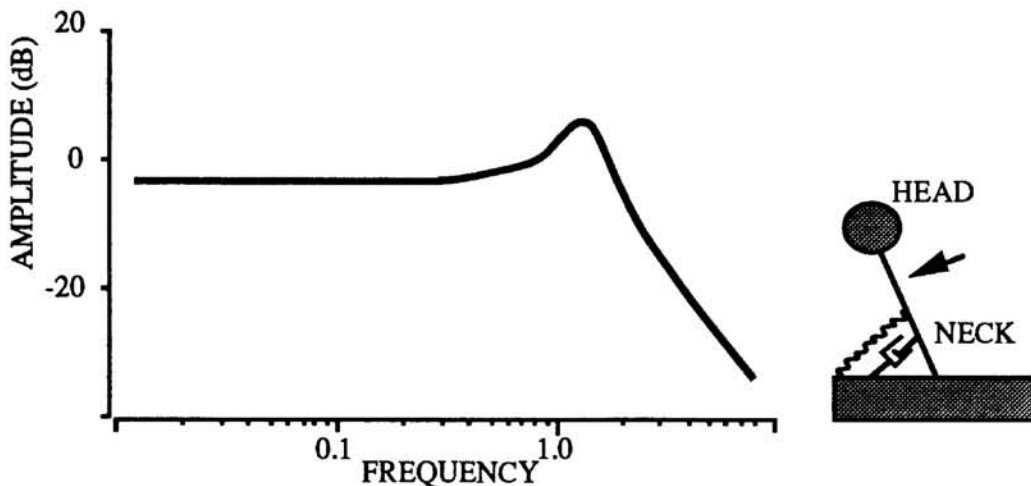

Figure 2: Amplitude spectrum of model head velocity.

We provide noisy measurements of "head" velocity and then ask what transfer function, or filter, will give the most accurate "eye" movement compensation? This is an estimation problem and, for Gaussian measurement error, the solution was discovered by Kalman and Bucy (1961). The optimal filter or estimator is often called the Kalman-Bucy filter. The gain and phase plots of the optimal filter for tracking movements of the inverted pendulum model are shown in figure 3. It can be seen that the gain of the optimal estimator for this system peaks near the maximum in the spectrum of "head-neck" velocity (fig. 2). This is a general feature of optimal filters. Accordingly, to accurately compensate for head movement in this system, the VOR would need to have a frequency dependent gain.

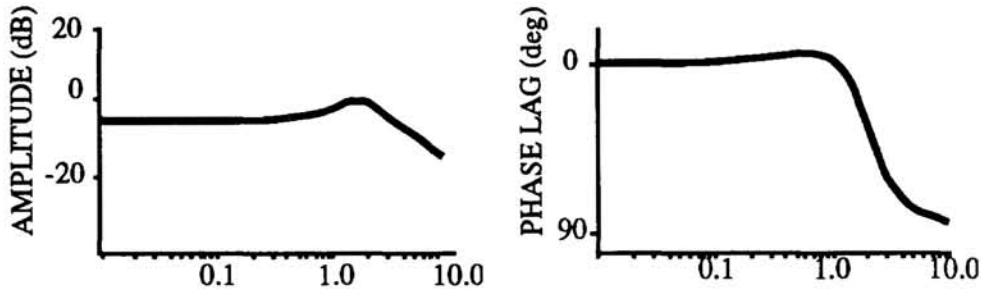

Figure 3: Bode gain plot (left) and phase plot (right) of an optimal estimator for tracking the inverted pendulum using noisy data.

**Time Varying dynamics and the VOR**

So far we have considered our model for VOR optimization only in the simple case of a constant head-neck velocity power spectrum. Under natural conditions, however, this spectrum would be expected to change. For example, when gait changes from walking to running, corresponding changes in the VOR transfer function would be necessary to maintain optimal performance. To explore this, we added a second inverted pendulum to our model to simulate body dynamics. We simulated changes in gait by changing the resonant frequency of the trunk. Figure 4 compares the spectra of head-neck velocity with two different trunk parameters. As in the previous example, we then computed transfer functions of the optimal filters for estimating head velocity from noisy measurements in these two cases. The gain and phase characteristics of these filters are also shown in Figure 5. These plots demonstrate that significant changes in the transfer function of the VOR would be necessary to maintain visual acuity in our model system under these different conditions. Of course, in the real situation head-neck dynamics will change rapidly and continuously with changes in gait, posture, substrate, etc. requiring rapid continuous changes in VOR dynamics rather than the simple switch implied here.

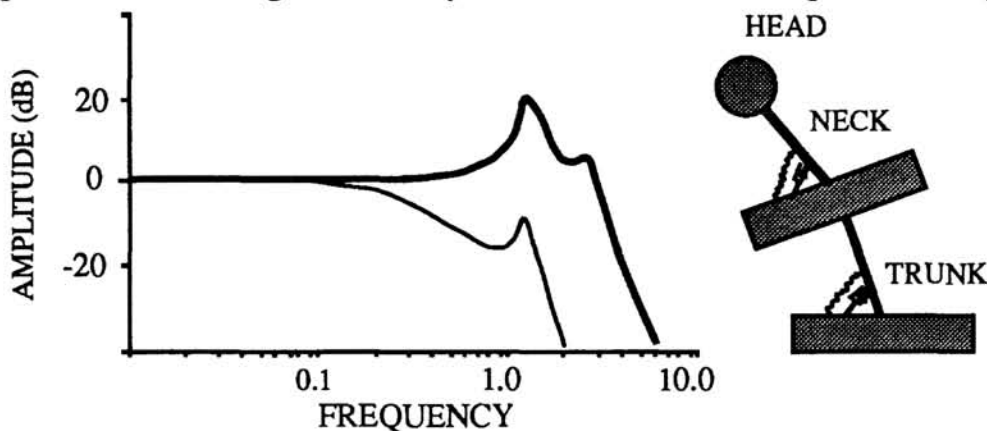

Figure 4: Head velocity spectrum during "walking" (light) and "running" (heavy).

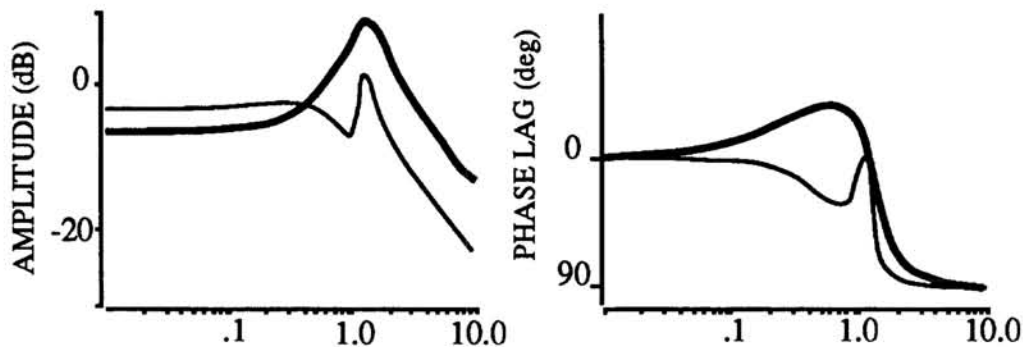

Figure 5: Bode gain plots (left) and phase plots (right) for optimal estimators of head angular velocity during "walking" (light) and "running" (heavy).

## 4. SIGNIFICANCE TO THE REAL VOR

Our results show that the optimal VOR transfer function requires a frequency dependent gain to accurately adjust to a wide range of head movements under real world conditions. Thus, the deviations from unity gain seen in actual measurements of the VOR may not represent poor, but rather optimal, performance. Our modeling similarly suggests that several other experimental results can be reinterpreted. For example, localized peaks or valleys in the VOR gain function can be induced experimentally through prolonged sinusoidal oscillations of subjects wearing magnifying or reducing lenses. However, this "frequency selectivity" is not thought to occur naturally and has been interpreted to imply the existence of frequency selective channels in the VOR control network (Lisberger, Miles and Optican, 1983). In our view there is no real distinction between this phenomenon and the "excessive" gain in normal monkey VOR; in each case the VOR optimizes its response for the particular task which it has to solve. This is testable. If we are correct, then frequency selective gain changes will occur following prolonged narrow-band rotation in the light *without wearing lenses*. In the classical framework there is no reason for any gain changes to occur in this situation.

Another phenomenon which has been observed experimentally and that the current modeling sheds new light on is referred to as "pattern storage". After single-frequency sinusoidal oscillation on a turntable in the light for several hours, rabbits will continue to produce oscillatory eye movements when the lights are extinguished and the turntable stops. Trained rabbits also produce eye oscillations at the training frequency when oscillated in the dark at a different frequency (Collewijn, 1985). In this case the sinusoidal pattern seems to be "stored" in the nervous system. However, the effect is naturally accounted for by our optimal estimator hypothesis without relying on an explicit "pattern storage mechanism". An optimal estimator works by matching its dynamics to the dynamics of the signal generator, and in effect it tries to force an internal model to mimic the signal generator by comparing actual and expected patterns of sensory inputs. When

no data is available, or the data is thought to be very unreliable, an optimal estimator relies completely, or almost completely, on the model. In cases where the signal is patterned the estimator will behave as though it had memorized the pattern. Thus, if we hypothesize that the VOR is an optimal estimator we do not need an extra hypothesis to explain pattern storage. Again, our hypothesis is testable. If we are correct, then repeating the pattern storage experiments using rotational velocity waveforms obtained by driving a frequency-tuned oscillator with Gaussian white noise will produce identical dynamical effects in the VOR. There is no sinusoidal pattern in the stimulus, but we predict that the rabbits can be induced to generate sinusoidal eye movements in the dark after this training.

The modeling results shown in figures 4 and 5 represent an extension of our ideas into the area of gait (or more generally "context") dependent changes in VOR which has not been considered very much in VOR research. In fact, VOR experimental paradigms, in general, are explicitly set up to produce the most stable VOR dynamics possible. Accordingly, little work has been done to quantify the short term changes in VOR dynamics that must occur in response to changes in effective head-neck dynamics. Experiments of this type would be valuable and are no more difficult technically than experiments which have already been done. For example, training an animal on a turntable which can be driven randomly with two distinct velocity power spectra, i.e. two "gaits", and providing the animal with external cues to indicate the gait would, we predict, result in an animal that could use the cues to switch its VOR dynamics. A more difficult but also more compelling demonstration would be to test VOR dynamics with impulsive head accelerations in different natural situations, using an unrestrained animal.

# 5. SENSOR FUSION AND PREDICTION

To this point, we have discussed compensatory eye movements by treating the VOR as a single input, single output system. This allowed us to concentrate on a particular aspect of VOR control: tracking a time-varying dynamical system (the head) using noisy data. In reality there are a number of other factors which make control of compensatory eye movements a somewhat more complex task than it appears to be when it is modeled using classical control theory. For example, a variety of vestibular as well as non-vestibular signals (e.g. visual, proprioceptive) relating to head movements are transmitted to the compensatory eye movement control network (Ito, 1984). This gives rise to a "sensor fusion" problem where data from different sources must be combined. The optimal solution to this problem for a multiple input - multiple output, time-varying linear, stochastic system is also given by the Kalman-Bucy filter (Davis and Vinter, 1985). Borah, Young and Curry (1988) have demonstrated that a Kalman-Bucy filter model of visual-vestibular sensor fusion is able to account for visual-vestibular interactions in motion perception. Oman (1982) has also developed a Kalman-Bucy filter model of visual-vestibular interactions. Their results show that the optimal estimation approach is useful for analyzing multivariate aspects of compensatory eye movement control, and complement our analysis of dynamical aspects.

Another set of problems arises in the VOR because of small time delays in neural transmission and muscle activation. To optimize its response, the mammalian VOR needs to make up for these delays by predicting head movements about 10msec in advance (ref). Once the dynamics of the signal generator have been identified, prediction can be performed using model-based estimation (Davis and Vinter, 1985). A neural analog of a Taylor series expansion has also been proposed as a model of prediction in the VOR (Pellionisz and Llinas, 1979), but this mechanism is extremely sensitive to noise in the data and was abandoned as a practical technique for general signal prediction several decades ago in favor of model-based techniques (Wiener, 1948). The later approach may be more appropriate for analyzing neural mechanisms of prediction (Arbib and Amari, 1985). An elementary description of optimal estimation theory for target tracking, and its possible relation to cerebellar function, is given by Paulin (1988).

# 6. ROLE OF CEREBELLAR CORTEX IN VOR CONTROL

To this point we have presented a novel characterization of the problem of compensatory eye movement control without considering the physical circuitry which implements the behavior. However, there are two parts to the optimal estimation problem. At each instant it is necessary to (a) filter the data using the optimal transfer function to drive the desired response and (b) determine what transfer function is optimal at that instant and adjust the filtering network accordingly. The first problem is fairly straightforward, and existing models of VOR demonstrate how a network of neurons based on known brainstem circuitry can implement a particular transfer function (Cannon and Robinson, 1985). The second problem is more difficult because requires continuous monitoring of the context in which head movements occur using a variety of sources of relevant data to tune the optimal filter for that context. We speculate that the cerebellar cortex performs this task.

First, the cortex of the vestibulo-cerebellum is in a position to make the required computation, since it receives detailed information from multiple sensory modalities that provide information on the state of the motor system (Ito, 1985). Second, the cerebellum projects to and appears to modulate the brain stem compensatory eye movement control network (Mackay and Murphy, 1979). We predict that the cerebellar cortex is necessary to produce rapid, context-dependent optimal state dependent changes in VOR transfer function which we have discussed. This speculation can be tested with turntable experiments similar to those described in section 4 above in the presence and absence of the cerebellar cortex.

## 7. THE GENERAL FUNCTION OF CEREBELLAR CORTEX

According to our hypothesis, the cerebellar cortex is required for making optimal compensatory eye movements during head movements. This is accomplished by continuously modifying the dynamics of the underlying control network in the brainstem, based on current sensory information. The function of the cerebellar cortex in this case can then be seen in a larger context as using primary sensory information (vestibular, visual) to coordinate the use of a motor system (the extraoccular eye muscles) to position a sensory array (the retina) to optimize the quality of sensory information available to the brain. We believe that this is the role played by the rest of the cerebellum for other sensory systems. Thus, we suspect that the hemispheres of the rat cerebellum, with their peri-oral tactile input (Bower et al., 1983), are involved in controlling the optimal use of these tactile surfaces in sensory exploration through the control of facial musculature. Similarly, the hemispheres of the primate cerebellum, which have hand and finger tactile inputs (Ito, 1984), may be involved in an analogous exploratory task in primates. These tactile sensory-motor systems are difficult to analyze, and we are currently studying a functionally analogous but more accessible model system, the electric sense of weakly electric fish (cf Rasnow et al., this volume).

## 8.CONCLUSION

Our view of the cerebellum assigns it an important dynamic role which contrasts markedly with the more limited role it was assumed to have in the past as a learning device (Marr, 1969; Albus, 1971; Robinson, 1976). There is evidence that cerebellar cortex has some learning abilities (Ito, 1984), but it is recognized that cerebellar cortex has an important dynamic role in motor control. However, there are widely differing opinions as to the nature of that role (Ito, 1985; Miles and Lisberger, 1981; Pellionisz and Llinas, 1979). Our proposal, that the VOR is a neural analog of an optimal estimator and that the cerebellar cortex monitors context and sets reflex dynamics accordingly, should not be interpreted as a claim that the nervous system actually implements the computations which are involved in applied optimal estimation, such as the Kalman-Bucy filter. Understanding the neural basis of cerebellar function will require the combined power of a number of experimental, theoretical and modeling approaches (cf Wilson et al., this volume). We believe that analyses of the kind presented here have an important role in characterizing behaviors controlled by the cerebellum.

**Acknowledgments**
This work was supported by the NIH (BNS 22205), the NSF (EET-8700064), and the Joseph Drown Foundation.

**References**
Arbib M.A. and Amari S. 1985. Sensori-moto Transformations in the Brain (with a critique of the tensor theory of the cerebellum). J. Theor. Biol. 112:123-155

Borah J., Young L.R. and Curry, R.E. 1988. Optimal Estimator Model for Human Spatial Orientation. In: Proc. N.Y. Acad. Sci. B. Cohen and V. Henn (eds.). In Press.

Bower J.M. and Woolston D.C. 1983. The Vertical Organization of Cerebellar Cortex. J. Neurophysiol. 49: 745-766.

Carpenter R.H.S. 1977. Movements of the Eyes. Pion, London.

Davis M.B.A. and Vinter R.B. 1985. Stochastic Modelling and Control. Chapman and Hall, NY.

Furman J.M., O'Leary D.P. and Wolfe J.W. 1982. Dynamic Range of the Frequency Response of the Horizontal Vestibulo-Ocular Reflex of the Alert Rhesus Monkey. Acta Otolaryngol. 93: 81

Ito, M. 1984. The Cerebellum and Neural Control. Raven Press, NY.

Kalman R.E. 1960. A New Approach to Linear Filtering and Prediction Problems. J. Basic Eng., March 1960.

Kalman R.E. and Bucy R.S. 1961. New Results in Linear Filtering and Prediction Theory. J. Basic Eng. , March 1961.

Lisberger, S.G. 1988. The Neural Basis for Learning of Simple Motor Skills. Science, 242:728-735.

Lisberger S.G. , Miles F.A. and Optican L.M. 1983. Frequency Selective Adaptation: Evidence for Channels in the Vestibulo-Ocular Reflex. J. Neurosci. 3:1234-1244

Mackay W.A. and Murphy J.T. 1979. Cerebellar Modulation of reflex Gain. Prog. Neurobiol. 13:361-417.

Oman C.M. 1982. A heuristic mathematical Model for the Dynamics of Sensory Conflict and Motion Sickness. Acta Oto-Laryngol. S392.

Paulin M.G. 1988. A Kalman Filter Model of the Cerebellum. In: Dynamic Interactions in Neural Networks: Models and Data. M. Arbib and S. Amari (eds). Springer-Verlag, NY. pp239-261.

Pellionisz A. and Llinas R. 1979. Brain Modelling by Tensor Network Theory and Computer Simulation. The Cerebellum: Distributed Processor for Predictive Coordination. Neuroscience 4:323-348.

Robinson D.A. 1976. Adaptive Control of the Vestibulo-Ocular Reflex by the Cerebellum. J. Neurophys. 36:954-969.

Robinson D.A. 1981. The Use of Control Systems Analysis in the Neurophysiology of Eye Movements. Ann. Rev. Neurosci. 4:463-503.

Wiener, N. 1948. Cybernetics: Communication and Control in the Animal and the Machine. MIT Press, Boston.
